# Convergence Properties of the K-Means Algorithms

**Léon Bottou**
Neuristique,
28 rue des Petites Ecuries,
75010 Paris, France
leon@neuristique.fr

**Yoshua Bengio**[*]
Dept. I.R.O.
Université de Montréal
Montreal, Qc H3C-3J7, Canada
bengioy@iro.umontreal.ca

## Abstract

This paper studies the convergence properties of the well known K-Means clustering algorithm. The K-Means algorithm can be described either as a gradient descent algorithm or by slightly extending the mathematics of the EM algorithm to this hard threshold case. We show that the K-Means algorithm actually minimizes the quantization error using the very fast Newton algorithm.

## 1 INTRODUCTION

K-Means is a popular clustering algorithm used in many applications, including the initialization of more computationally expensive algorithms (Gaussian mixtures, Radial Basis Functions, Learning Vector Quantization and some Hidden Markov Models). The practice of this initialization procedure often gives the frustrating feeling that K-Means performs most of the task in a small fraction of the overall time. This motivated us to better understand this convergence speed.

A second reason lies in the traditional debate between hard threshold (e.g. K-Means, Viterbi Training) and soft threshold (e.g. Gaussian Mixtures, Baum Welch) algorithms (Nowlan, 1991). Soft threshold algorithms are often preferred because they have an elegant probabilistic framework and a general optimization algorithm named EM (expectation-maximization) (Dempster, Laird and Rubin, 1977). In the case of a gaussian mixture, the EM algorithm has recently been shown to *approximate* the Newton optimization algorithm (Xu and Jordan, 1994). We prove in this

---

[*]also, AT&T Bell Labs, Holmdel, NJ 07733

paper that the corresponding hard threshold algorithm, K-Means, minimizes the quantization error using *exactly* the Newton algorithm.

In the next section, we derive the K-Means algorithm as a gradient descent procedure. Section 3 extends the mathematics of the EM algorithm to the case of K-Means. This second derivation of K-Means provides us with proper values for the learning rates. In section 4 we show that this choice of learning rates optimally rescales the parameter space using Newton's method. Finally, in section 5 we present and discuss a few experimental results comparing various versions of the K-Means algorithm. The 5 clustering algorithms presented here were chosen for a good coverage of the algorithms related to K-Means, but this paper does not have the ambition of presenting a literature survey on the subject.

## 2   K-MEANS AS A GRADIENT DESCENT

Given a set of $P$ examples $(x_i)$, the K-Means algorithm computes $k$ prototypes $w = (w_k)$ which minimize the *quantization error*, i.e., the average distance between each pattern and the closest prototype:

$$E(w) \stackrel{\text{def}}{=} \sum_i L(x_i, w) \stackrel{\text{def}}{=} \sum_i \frac{1}{2} \min_k (x_i - w_k)^2 \qquad (1)$$

Writing $s_i(w)$ for the subscript of the closest prototype to example $x_i$, we have

$$E(w) = \sum_i \frac{1}{2}(x_i - w_{s_i(w)})^2 \qquad (2)$$

### 2.1   GRADIENT DESCENT ALGORITHM

We can then derive a *gradient descent* algorithm for the quantization error: $\Delta w = -\epsilon_t \frac{\partial E(w)}{\partial w}$. This leads to the following *batch update* equation (updating prototypes after presenting all the examples):

$$\Delta w_k = \sum_i \left\{ \begin{array}{ll} \epsilon_t(x_i - w_k) & \text{if } k = s_i(w) \\ 0 & \text{otherwise.} \end{array} \right. \qquad (3)$$

We can also derive a corresponding *online* algorithm which updates the prototypes after the presentation of each pattern $x_i$:

$$\Delta w = -\epsilon_t \frac{\partial L(x_i, w)}{\partial w}, \quad \text{i.e.,}$$

$$\Delta w_k = \left\{ \begin{array}{ll} \epsilon_t(x_i - w_k) & \text{if } k = s_i(w) \\ 0 & \text{otherwise.} \end{array} \right. \qquad (4)$$

The proper value of the learning rate $\epsilon_t$ remain to be specified in both batch and online algorithms. Convergence proofs for both algorithms (Bottou, 1991) exist for decreasing values of the learning rates satisfying the conditions $\sum \epsilon_t = \infty$ and $\sum \epsilon_t^2 < \infty$. Following (Kohonen, 1989), we could choose $\epsilon_t = \epsilon_0/t$. We prove however in this paper that there exist a much better choice of learning rates.

# 3  K-MEANS AS AN EM STYLE ALGORITHM

## 3.1  EM STYLE ALGORITHM

The following derivation of K-Means is similar to the derivation of (MacQueen, 1967). We insist however on the identity between this derivation and the mathematics of EM (Liporace, 1976) (Dempster, Laird and Rubin, 1977).

*Although K-Means does not fit in a probabilistic framework*, this similarity holds for a very deep reason: The semi-ring of probabilies $(\Re^+, +, \times)$ and the idempotent semi-ring of hard-threshold scores $(\Re, \text{Min}, +)$ share the most significant algebraic properties (Bacceli, Cohen and Olsder, 1992). This assertion completely describes the similarities and the potential differences between soft-threshold and hard-threshold algorithms. A complete discussion however stands outside the scope of this paper.

The principle of EM is to introduce additional "hidden" variables whose knowledge would make the optimization problem easier. Since these hidden variables are unknown, we maximize an auxiliary function which averages over the values of the hidden variables given the values of the parameters at the previous iteration. In our case, the hidden variables are the assignments $s_i(w)$ of the patterns to the prototypes. Instead of considering the expected value over the distribution on these hidden variables, we just consider the values of the hidden variables that minimize the cost, given the previous values of the parameters:

$$Q(w', w) \overset{\text{def}}{=} \sum_i \frac{1}{2}(x_i - w'_{s_i(w)})^2$$

The next step consists then in finding a new set of prototypes $w'$ which minimizes $Q(w', w)$ where $w$ is the previous set of prototypes. We can analytically compute the explicit solution of this minimization problem. Solving the equation $\partial Q(w', w)/\partial w'_k = 0$ yields:

$$w'_k = \frac{1}{N_k} \sum_{i:k=s_i(w)} x_i \tag{5}$$

where $N_k$ is the number of examples assigned to prototype $w_k$. The algorithm consists in repeatedly replacing $w$ by $w'$ using update equation (6) until convergence. Since $s_i(w')$ is by definition the best assignment of patterns $x_i$ to the prototypes $w'_k$, we have the following inequality:

$$E(w') - Q(w', w) = \frac{1}{2} \sum_i (x_i - w'_{s_i(w')})^2 - (x_i - w'_{s_i(w)})^2 \le 0$$

Using this result, the identity $E(w) = Q(w, w)$ and the definition of $w'$, we can derive the following inequality:

$$
\begin{aligned}
E(w') - E(w) &= E(w') - Q(w', w) + Q(w', w) - Q(w, w) \\
&\le Q(w', w) - Q(w, w) \le 0
\end{aligned}
$$

Each iteration of the algorithm thus decreases the otherwise positive quantization error $E$ (equation 1) until the error reaches a fixed point where condition $w^{*\prime} = w^*$ is verified (unicity of the minimum of $Q(\bullet, w^*)$). Since the assignment functions $s_i(w)$ are discrete, there is an open neighborhood of $w^*$ on which the assignments are constant. According to their definition, functions $E(\bullet)$ and $Q(\bullet, w^*)$ are equal on this neighborhood. Being the minimum of function $Q(\bullet, w^*)$, the fixed point $w^*$ of this algorithm is also a local minimum of the quantization error $E$.  □

## 3.2  BATCH K-MEANS

The above algorithm (5) can be rewritten in a form similar to that of the batch gradient descent algorithm (3).

$$\Delta w_k = w'_k - w_k = \sum_i \left\{ \begin{array}{ll} \frac{1}{N_k}(x_i - w_k) & \text{if } k = s(x_i, w) \\ 0 & \text{otherwise.} \end{array} \right. \qquad (6)$$

This algorithm is thus equivalent to a batch gradient descent with a specific, prototype dependent, learning rate $\frac{1}{N_k}$.

## 3.3  ONLINE K-MEANS

The online version of our EM style update equation (5) is based on the computation of the mean $\mu_t$ of the examples $x_1, \cdots, x_t$ with the following recursive formula:

$$\mu_{t+1} = \frac{1}{t+1}(t\,\mu_t + x_{t+1}) = \mu_t + \frac{1}{t+1}(x_{t+1} - \mu_t)$$

Let us introduce new variables $n_k$ which count the number of examples so far assigned to prototype $w_k$. We can then rewrite (5) as an online update applied after the presentation of each pattern $x_i$:

$$\begin{aligned} \Delta n_k &= \left\{ \begin{array}{ll} 1 & \text{if } k = s(x_i, w) \\ 0 & \text{otherwise.} \end{array} \right. \\ \Delta w_k &= \left\{ \begin{array}{ll} \frac{1}{n_k}(x_i - w_k) & \text{if } k = s(x_i, w) \\ 0 & \text{otherwise.} \end{array} \right. \end{aligned} \qquad (7)$$

This algorithm is equivalent to an online gradient descent (4) with a specific, prototype dependent, learning rate $\frac{1}{n_k}$. Unlike in the batch case, the pattern assignments $s(x_i, w)$ are thus changing after each pattern presentation. Before applying this algorithm, we must of course set $n_k$ to zero and $w_k$ to some initial value. Various methods have been proposed including initializing $w_k$ with the first $k$ patterns.

## 3.4  CONVERGENCE

General convergence proofs for the batch and online gradient descent (Bottou, 1991; Driancourt, 1994) directly apply for all four algorithms. Although the derivatives are undefined on a few points, these theorems prove that the algorithms almost surely converge to a local minimum because the local variations of the loss function are conveniently bounded (semi-differentiability). Unlike previous results, the above convergence proofs allow for non-linearity, non-differentiability (on a few points) (Bottou, 1991), and replacing learning rates by a positive definite matrix (Driancourt, 1994).

# 4  K-MEANS AS A NEWTON OPTIMIZATION

We prove in this section that Batch K-Means (6) applies the Newton algorithm.

## 4.1  THE HESSIAN OF K-MEANS

Let us compute the Hessian $H$ of the K-Means cost function (2). This matrix contains the second derivatives of the cost $E(w)$ with respect to each pair of parameters. Since $E(w)$ is a sum of terms $L(x_i, w)$, we can decompose $H$ as the sum

of matrices $H_i$ for each term of the cost function:

$$L(x_i, w) = \min_k \frac{1}{2}(x_i - w_k)^2.$$

Furthermore, the $H_i$ can be decomposed in blocks corresponding to each pair of prototypes. Since $L(x_i, w)$ depends only on the closest prototype to pattern $x_i$, all these blocks are zero except block $(s_i(w), s_i(w))$ which is the identity matrix. Summing the partial Hessian matrices $H_i$ thus gives a diagonal matrix whose diagonal elements are the counts of examples $N_k$ assigned to each prototype.

$$H = \begin{pmatrix} N_1 I & 0 & \cdots & 0 \\ 0 & N_2 I & \cdots & 0 \\ \vdots & \vdots & & \vdots \\ 0 & 0 & \cdots & N_K I \end{pmatrix}$$

We can thus write the Newton update of the parameters as follows:

$$\Delta w = -H^{-1} \frac{\partial E(w)}{\partial w}$$

which can be exactly rewritten as the batch EM style algorithm (6) presented earlier:

$$\Delta w_k = \sum_i \left\{ \begin{array}{ll} \frac{1}{N_k}(x_i - w_k) & \text{if } k = s(x_i, w) \\ 0 & \text{otherwise.} \end{array} \right. \tag{8}$$

## 4.2 CONVERGENCE SPEED

When optimizing a quadratic function, Newton's algorithm requires only one step. In the case of a non quadratic function, Newton's algorithm is superlinear if we can bound the variations of the second derivatives. Standard theorems that bound this variation using the third derivative are not useful for K-Means because the gradient of the cost function is discontinuous. We could notice that the variations of the second derivatives are however nicely bounded and derive similar proofs for K-Means.

For the sake of brevity however, we are just giving here an intuitive argument: we can make the cost function indefinitely differentiable by rounding up the angles around the non differentiable points. We can even restrict this cost function change to an arbitrary small region of the space. The iterations of K-Means will avoid this region with a probability arbitrarily close to 1. In practice, we obtain thus a superlinear convergence.

Batch K-Means thus searches for the optimal prototypes at Newton speed. Once it comes close enough to the optimal prototypes (i.e. the pattern assignment is optimal and the cost function becomes quadratic), K-Means jumps to the optimum and terminates.

Online K-Means benefits of these optimal learning rates because they remove the usual conditioning problems of the optimization. However, the stochastic noise induced by the online procedure limits the final convergence of the algorithm. Final convergence speed is thus essentially determined by the schedule of the learning rates.

Online K-Means also benefits from the redundancies of the training set. It converges significantly faster than batch K-Means during the first training epochs (Darken

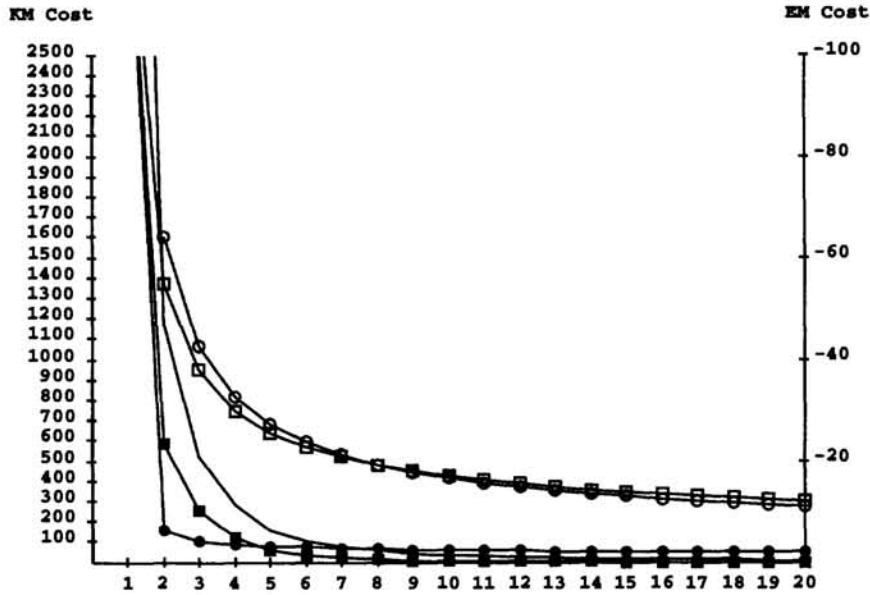

Figure 1: $E_t - E_\infty$ versus $t$. black circles: online K-Means; black squares: batch K-Means; empty circles: online gradient; empty squares: batch gradient; no mark: EM+Gaussian mixture

and Moody, 1991). After going through the first few patterns (depending of the amount of redundancy), online K-Means indeed improves the prototypes as much as a complete batch K-Means epoch. Other researchers have compared batch and online algorithms for neural networks, with similar conclusions (Bengio, 1991).

## 5  EXPERIMENTS

Experiments have been carried out with Fisher's iris data set, which is composed of 150 points in a four dimensional space representing physical measurements on various species of iris flowers. Codebooks of six prototypes have been computed using both batch and online K-Means with the proper learning rates (6) and (7). These results are compared with those obtained using both gradient descent algorithms (3) and (4) using learning rate $\epsilon_t = 0.03/t$ that we have found optimal. Results are also compared with likelihood maximization with the EM algorithm, applied to a mixture of six Gaussians, with fixed and uniform mixture weights, and fixed unit variance. Inputs were scaled down empirically so that the average cluster variance was around unity. Thus only the cluster positions were learned, as for the K-Means algorithms.

Each run of an algorithm consists in (a) selecting a random initial set of prototypes, (b) running the algorithm during 20 epochs and recording the error measure $E_t$ after each epoch, (c) running the batch K-Means algorithm[1] during 40 more epochs in order to locate the local minimum $E_\infty$ corresponding to the current initialization of the algorithm. For the four K-Means algorithms, $E_t$ is the quantization error (equation 1). For the Gaussian mixture trained with EM, the cost $E_t$ is the negative

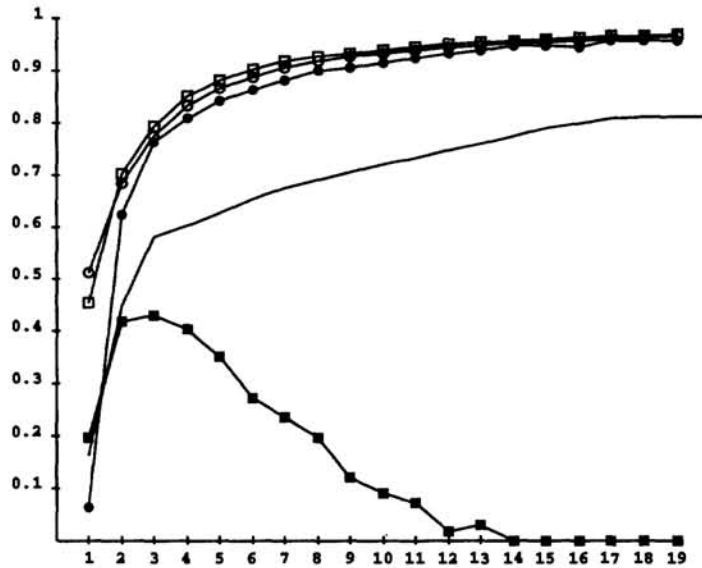

Figure 2: $\frac{E_{t+1}-E_\infty}{E_t-E_\infty}$ versus $t$. black circles: online K-Means; black squares: batch K-Means; empty circles: online gradient; empty squares: batch gradient; no mark: EM+Gaussian mixture

logarithm of the likelihood of the data given the model.

Twenty trials were run for each algorithm. Using more than twenty runs did not improve the standard deviation of the averaged measures because various initializations lead to very different local minima. The value $E_\infty$ of the quantization error on the local minima ranges between 3300 and 5800. This variability is caused by the different initializations and not by the different algorithms. The average values of $E_\infty$ for each algorithm indeed fall in a very small range (4050 to 4080).

Figure 1 shows the average value of the residual error $E_t - E_\infty$ during the first 20 epochs. Online K-Means (black circles) outperforms all other algorithms during the first five epochs and stabilizes on a level related to the stochastic noise of the online procedure. Batch K-Means (black squares) initially converges more slowly but outperforms all other methods after 5 epochs. All 20 runs converged before the 15th epoch. Both gradients algorithms display poor convergence because they do not benefit of the Newton effect. Again, the online version (white circles) starts faster then the batch version (white square) but is outperformed in the long run. The negative logarithm of the Gaussian mixture is shown on the curve with no point marks, and the scale is displayed on the right of Figure 1.

Figure 2 show the final convergence properties of all five algorithms. The evolutions of the ratio $(E_{t+1} - E_\infty)/(E_t - E_\infty)$ characterize the relative improvement of the residual error after each iteration. All algorithms exhibit the same behavior after a few epochs except batch K-Means (black squares). The fast convergence of this ratio to zero demonstrates the final convergence of batch K-Means. The EM algorithm displays a better behavior than all the other algorithms except batch K-Means. Clearly, however, its relative improvement ratio doesn't display the fast convergence behavior of batch K-Means.

The online K-Means curve crosses the batch K-Means curve during the second epoch, suggesting that it is better to run the online algorithm (7) during one epoch and then switch to the batch algorithm (6).

## 6   CONCLUSION

We have shown with theoretical arguments and simple experiments that a well implemented K-Means algorithm minimizes the quantization error using Newton's algorithm. The EM style derivation of K-Means shows that the mathematics of EM are valid well outside the framework of probabilistic models. Moreover the provable convergence properties of the hard threshold K-Means algorithm are superior to those of the EM algorithm for an equivalent soft threshold mixture of Gaussians. Extending these results to other hard threshold algorithms (e.g. Viterbi Training) is an interesting open question.

## Footnotes

[1] except for the case of the mixture of Gaussians, in which the EM algorithm was applied

## References

Bacceli, F., Cohen, G., and Olsder, G. J. (1992). *Synchronization and Linearity.* Wiley.

Bengio, Y. (1991). *Artificial Neural Networks and their Application to Sequence Recognition.* PhD thesis, McGill University, (Computer Science), Montreal, Qc., Canada.

Bottou, L. (1991). *Une approche théorique de l'apprentissage connexioniste; applications à la reconnaissance de la parole.* PhD thesis, Université de Paris XI.

Darken, C. and Moody, J. (1991). Note on learning rate schedules for stochastic optimization. In Lippman, R. P., Moody, R., and Touretzky, D. S., editors, *Advances in Neural Information Processing Systems 3*, pages 832–838, Denver, CO. Morgan Kaufmann, Palo Alto.

Dempster, A. P., Laird, N. M., and Rubin, D. B. (1977). Maximum-likelihood from incomplete data via the EM algorithm. *Journal of Royal Statistical Society B*, 39:1–38.

Driancourt, X. (1994). *Optimisation par descente de gradient stochastique ....* PhD thesis, Université de Paris XI, 91405 Orsay cedex, France.

Kohonen, T. (1989). *Self-Organization and Associative Memory.* Springer-Verlag, Berlin, 3 edition.

Liporace, L. A. (1976). PTAH on continuous multivariate functions of Markov chains. Technical Report 80193, Institute for Defense Analysis, Communication Research Department.

MacQueen, J. (1967). Some methods for classification and analysis of multivariate observations. In *Proceedings of the Fifth Berkeley Symposium on Mathematics, Statistics and Probability, Vol. 1*, pages 281–296.

Nowlan, S. J. (1991). *Soft Competitive Adaptation: Neural Network Learning Algorithms based on Fitting Statistical Mixtures.* CMU-CS-91-126, School of Computer Science, Carnegie Mellon University, Pittsburgh, PA.

Xu, L. and Jordan, M. (1994). Theoretical and experimental studies of convergence properties of the em algorithm for unsupervised learning based on finite mixtures. Presented at the Neural Networks for Computing Conference.